# Unsupervised Pixel-prediction

**William R. Softky**
Math Research Branch
NIDDK, NIH
9190 Wisconsin Ave #350
Bethesda, MD 20814
bill@homer.niddk.nih.gov

## Abstract

When a sensory system constructs a model of the environment from its input, it might need to verify the model's accuracy. One method of verification is multivariate time-series prediction: a good model could predict the near-future activity of its inputs, much as a good scientific theory predicts future data. Such a predicting model would require copious top-down connections to compare the predictions with the input. That feedback could improve the model's performance in two ways: by biasing internal activity toward expected patterns, and by generating specific error signals if the predictions fail. A proof-of-concept model—an event-driven, computationally efficient layered network, incorporating "cortical" features like all-excitatory synapses and local inhibition—was constructed to make near-future predictions of a simple, moving stimulus. After unsupervised learning, the network contained units not only tuned to obvious features of the stimulus like contour orientation and motion, but also to contour discontinuity ("end-stopping") and illusory contours.

## 1 Introduction

Somehow, brains make very accurate models of the outside world from their raw sensory input. How might brains check and improve those models? What signal is there to verify a model of the world?

The scientific method faces a similar problem: how to verify theories. In science, theories are verified by predicting *future* data, using the implicit assumption that

good predictions can only result from good models. By analogy, it is possible that brains predict their afferent input (e.g. at the thalamus), and that making such predictions and using them as feedback is a unifying design principle of cortex. The proof-of-concept model presented here uses unsupervised Hebbian learning to predict, pixel-wise, the location of a moving pattern slightly in the future.

Why try prediction?

• Predicting future data usually requires a good generative model. For instance: to predict the brightness of individual TV pixels even a fraction of a second in advance, one would need models of contours, objects, motion, occlusion, shadow, etc.

• A successful prediction can help filter out input noise, like a Kalman filter.

• A failed prediction provides a specific, high-dimensional error signal.

• Prediction is not only possible in cortex—which has massive feedback connections—but necessary as well, because those feedback fibers, their target dendrites, and synaptic integration impose inevitable delays. So for a feedback signal to arrive at the cell body "on time," it would need to have been generated tens of milliseconds earlier, as a prediction of imminent activity.

• In this model, "prediction" means producing spikes in advance which will correlate with subsequent input spikes. Specifically, the network's goal is to produce at each grid point a train of spikes at times $P_j$ which predicts the input train $I_k$, in the sense of maximizing their normalized cross-correlation. The objective function $L$ ("likeness") can be expressed in terms of a smoothing "bump" function $B(t_x, t_y)$ (of spikes at times $t_x$ and $t_y$) and a correlation function $C(train_1, train_2, \Delta t)$:

$$B(t_x, t_y) = \exp\left(\frac{-|t_x - t_y|}{\tau}\right)$$
$$C(P, I, \Delta T) = \sum_j \sum_k B(P_j + \Delta t, I_k)$$
$$L(P, I, \Delta T) = \frac{C(P, I, \Delta T)}{\sqrt{C(P, P, 0)C(I, I, 0)}}$$

• In order to avoid a trivial but useless prediction ("the weather tomorrow will be just like today"), one must ensure that a unit cannot usually predict its own firing (for example, pick $\Delta t \approx \tau$ greater than the autocorrelation time of a spike train).

## 2  Model

The input to the network is a $16 \times 16$ array of spike trains, with toroidal array boundary conditions. The spikes are driven by a "stimulus" bar of excitation one unit wide and seven units long, which moves smoothly perpendicular to its orientation behind the array (in a broad circle, so that all orientations and directions are represented; Fig. 1A). The stimulus point transiently generates spikes at each grid point there according to a Poisson process: the whole array of spikes can be visualized as a twinkling, moving contour.

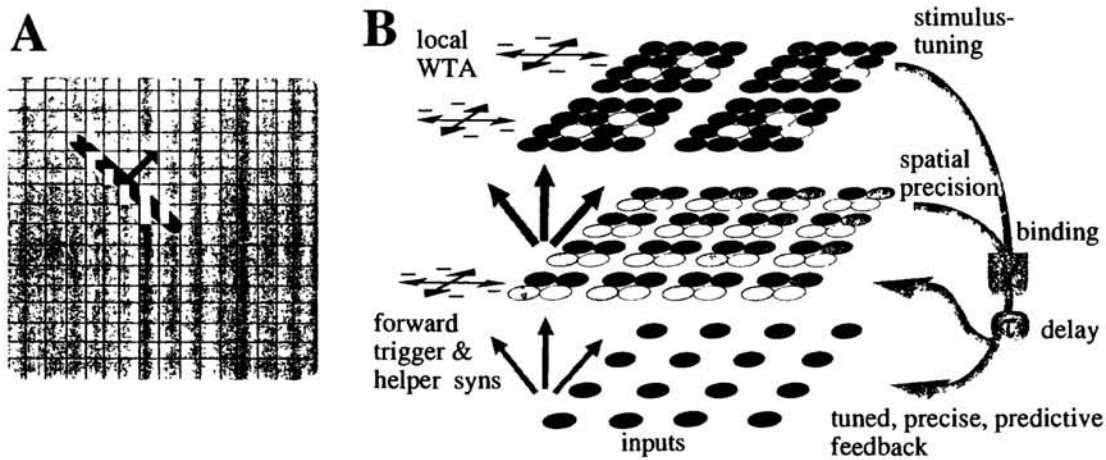

Figure 1: **A network predicts dynamic patterns.** **A** A moving pattern on a grid of spiking pixels describes a slow circle, and drives activity in a network above. **B** The three-layer network learns to predict that activity just before it occurs. Forward connections, evolving by Hebbian rules, produce top-level units with coarse receptive fields and fine stimulus-tuning (e.g. contour orientation and motion). Each spike from a top unit is "bound" (by coincidence detection) with the particular spike which triggered it, to produce feedback which is both stimulus-tuned and spatially specific. A Hebb rule determines how the delayed, predictive feedback will drive middle-layer units and be compared to input-layer units. Because all connections are excitatory, winner-take-all inhibition within local groups of units prevents runaway excitation.

## 2.1 Network Structure

The network has three layers. The bottom layer contains the spiking pixels, and the "surprise" units described below. The middle layer, having the same spatial resolution as the input, has four coarsely-tuned units per input pixel. And the top layer contains the most finely-tuned units, spaced at half the spatial resolution (at every fourth gridpoint, i.e. with coarser spatial resolution and larger receptive fields). The signal flow is bi-directional [10, 7], with both forward and feedback synaptic connections. All connections between units are excitatory, and excitation is kept in check by local winner-take-all inhibition (WTA). For example, a given input spike can only trigger one spike out of the 16 units directly above it in the top layer (Fig. 1B).

Unsupervised learning occurs through two local Hebb-like rules. Forward connections evolve to make nearby (competing) units strongly anticorrelated—for instance, units typically become tuned to different contour orientations and directions of motion—while feedback connections evolve to maximally correlate delayed feedback signals with their targets.

## 2.2 Binary multiplication in single units

While some neural models implement multiplication as a nonlinear function of the sum of the inputs, the spiking model used here implements multiplication as a binary operation on two distinct classes of synapses.

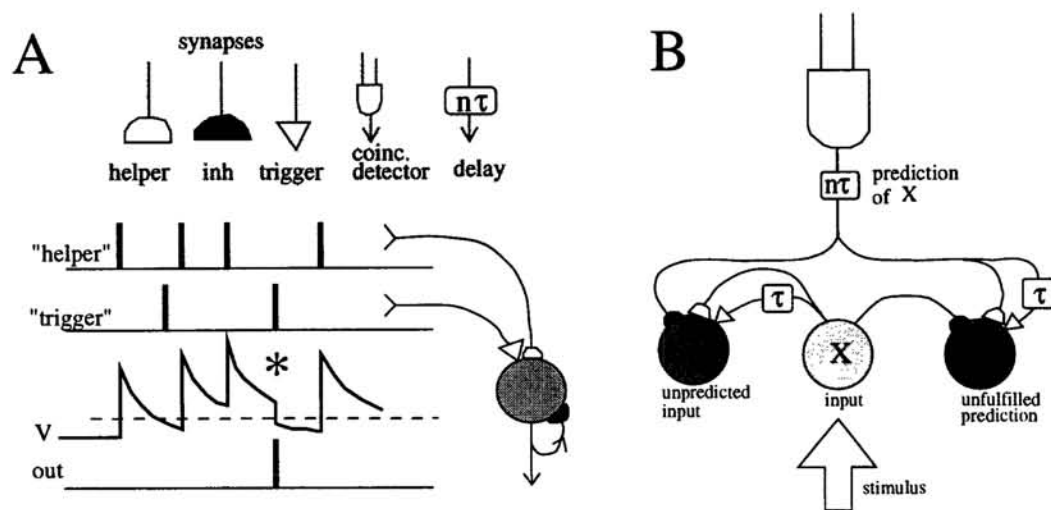

Figure 2: **Multiplicative synapses and surprise detection.** **A** A spiking unit
multiplies two types of synaptic inputs: the "helper" type increments an internal
bias without triggering a spike, and the "trigger" type can trigger a spike (*),
without incrementing, but only if the bias is above a threshold. Spike propagation
may be discretely delayed, and coincidences of two units fired by the same input
spike can be detected. **B** Once the network has generated a (delayed) prediction of
a given pixel's activity, the match of prediction and reality can be tested by special-
purpose units: one type which detects unpredicted input, the other which detects
unfulfilled predictions. The firing of either type can drive the network's learning
rules, so units above can become tuned to consistent patters of failed predictions,
as occur at discontinuities and illusory contours.

A *helper* synapse, when activated by a presynaptic spike, will increment or decre-
ment the postsynaptic voltage without ever initiating a spike. A *trigger* synapse, on
the other hand, can initiate a spike (if the voltage is above the threshold determined
by its WTA neighbors), but cannot adjust the voltage (Fig. 2A; the helper type is
loosely based on the weak, slow NMDA synapses on cortical apical dendrites, while
triggers are based on strong, brief AMPA synapses on basal dendrites.) Thus, a
unit can only fire when both synaptic types are active, so the output firing rate
approximates the product of the rates of helpers and triggers. Each unit has two
characteristic timescales: a slower voltage decay time, and the essentially instanta-
neous time necessary to trigger and propagate a spike.

This scheme has two advantages. One is that a single cell can implement a relatively
"pure" multiplication of distinct inputs, as required for computations like motion-
detection. The other advantage is that feedback signals, restricted to only helper
synapses, cannot by themselves drive a cell, so closed positive-feedback loops cannot
"latch" the network into a fixed state, independent of the input. Therefore, all
trigger synapses in this network are forward, while all delayed, lateral, and feedback
connections are of the helper type.

## 2.3   Feedback

There are two issues in feedback: How to construct tuned, specific feedback, and
what to do with the feedback where it arrives.

An accurate prediction requires information about the input: both about its exact present state, and about its history over nearby space and recent time. In this model, those signals are distinct: spatial and temporal specificity is given by each input spike, and the spatio-temporal history is given by the stimulus-tuned responses of the slow, coarse-grained units in the top layer. Spatially-precise feedback requires recombining those signals. (Feedback from V1 cortical Layer VI to thalamus has recently been shown to fit these criteria, being both spatially refined and direction-selective; [3] Grieve & Sillito, 1995).

In this network, each feedback signal results from the AND of spikes from a input-layer spike (spatially specific) and the resulting top-layer spike it produces (stimulus-tuned). This "binding" across levels of specificity requires single-spike temporal precision, and may even be one of the perceptual uses for spike timing in cortex [1, 9].

## 2.4   Surprise detection

Once predictive feedback is learned, it can be used in two ways: biasing units toward expected activity, and comparing predictions against actual input. Feedback to the middle layer is used as a bias signal through *helper* synapses, by adding the feedback to the bias signal. But feedback to the bottom, input-layer is compared with actual input by means of special "surprise" units which subtract prediction from input (and vice versa).

Because both prediction and input are noisy signals, their difference is even noisier, and must be both temporally smoothed and thresholded to generate a mismatch-spike. In this model, these prediction/input differences are accomplished pixel-by-pixel using ad-hoc units designed for the purpose (Fig. 2B). There is no indication that cortex operates so simplistically, but there are indications that cortical cells are in general sensitive to mismatches between expectation and reality, such as discontinuities in space (edges), in time (on- and off-responses), and in context (saliency).

The resulting error vector can drive upper-layer units just as the input does, so that the network can learn patterns of failed predictions, which typically correspond to discontinuities in the stimulus. Learning consistent patterns of bad predictions is a completely generic recipe for discovering such discontinuitites, which often correspond closely to visually important features like contour ends, corners, illusory contours, and occlusion.

## 3   Results and Discussion

After prolonged exposure to the stimulus, the network produces a blurred cloud of spikes which anticipates the actual input spikes, but which also consistently predicts input beyond the bar's ends (leading to small clouds of surprise-unit activity tracking the ends). The top-level units, driven both by input signals and by feedback, become tuned either to different motions of the bar itself (due to Hebbian learning of the input), or to different motions of its *ends* (due to Hebbian learning of the surprise-units); see Fig. 3. Cells tuned to contour ends ("end-stopped") have been found in visual cortex [11], although the principles of their genesis are not known. Using the same parameters but a different stimulus, the network can also evolve units

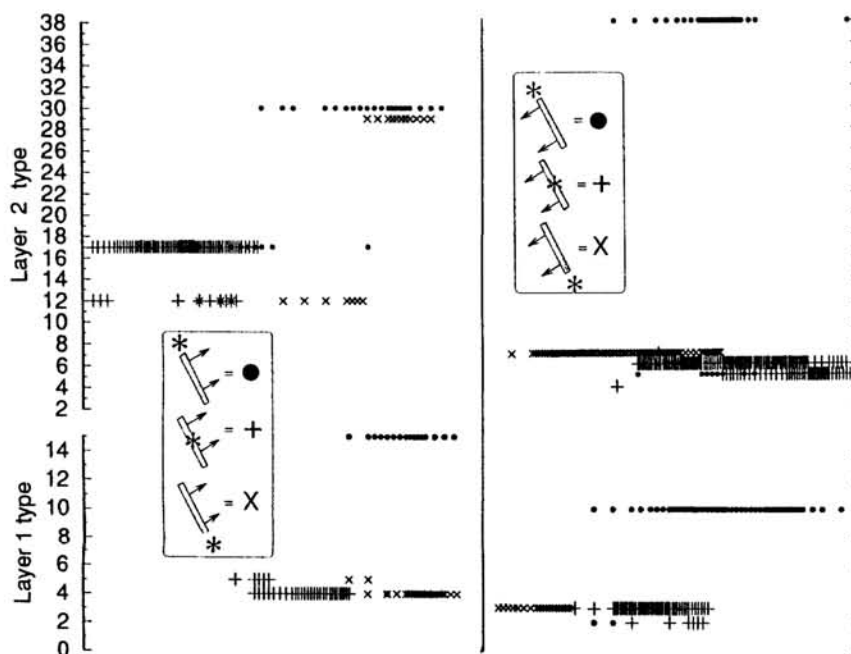

Figure 3: **Single units are highly stimulus-specific.** Spikes from all units at
one location are shown (with time) as a stimulus bar (insets) passes them with six
different relative positions and motions. Out of the many units available, only one
or two are active in each layer for a given stimulus configuration. The inactive
units are tuned to stimulus orientations not shown here. Some units are driven by
"surprise" units (Figure 2 and text), and respond only to the bar's ends (• and ×),
but not to its center (+). Such responses lag behind those of ordinary units, because
they must temporally integrate to determine whether a significant mismatch exists
between the noisy prediction and the noisy input. Spikes from five passes have been
summed to show the units' reliability.

which detect the illusory contours present in certain moving gratings.

Several researchers propose that cortex (or similar networks) might use feedback
pathways to recreate or regenerate their (static) input [7, 4, 10]. The approach here
requires instead that the network forecast future (dynamic) input [8]. In a general
sense, predicting the future is a better test of a model than predicting the present,
in the same sense that scientific theories which predict future experimental data are
more persuasive than theories which predict existing data. Prediction of the raw
input has advantages over prediction of some higher-level signal [5, 6, 2]: the raw
input is the only unprocessed "reality" available to the network, and comparing the
prediction with that raw input yields the highest-dimensional error vector possible.

Spiking networks are likewise useful. As in cortex, spikes both truncate small inputs
and contaminate them with quantization-noise, crucial practical problems which
real-valued networks avoid. Spike-driven units can implement purely correlative
computations like motion-detection, and can avoid parasitic positive-feedback loops.
Spike timing can identify which of many possible inputs fired a given unit, thereby
making possible a more specific feedback signal. The most practical benefit is that
interactions among rare events (like spikes) are much faster to compute than real-

valued ones; this particular network of 8000 units and 200,000 synapses runs faster than the workstation can display it.

This model is an ad-hoc network to illustrate some of the issues a brain might face in trying to predict its retinal inputs; it is not a model of cortex. Unfortunately, the hypothesis that cortex predicts its own inputs does not suggest any specific circuit or model to test. But two experimental tests may be sufficiently model-independent. One is that cortical "non-classical" receptive fields should have a temporal structure which reflects the temporal sequences of natural stimuli, so a given cell's activity will be either enhanced or suppressed when its input matches contextual expectations. Another is that feedback to a single cell in thalamus, or to an individual cortical apical dendrite, should arrive on average *earlier* than afferent input to the same cell.

# References

[1] A. Engel, P. Koenig, A. Kreiter, T. Schillen, and W. Singer. Temporal coding in the visual cortex: New vistas on integration in the nervous system. *TINS*, 15:218–226, 1992.

[2] K. Fielding and D. Ruck. Recognition of moving light displays using hidden markov models. *Pattern Recognition*, 28:1415–1421, 1995.

[3] K. L. Grieve and A. M. Sillito. Differential properties of cells in the feline primary visual cortex providing the cortifugal feedback to the lateral geniculate nucleus and visual claustrum. *J. Neurosci.*, 15:4868–4874, 1995.

[4] G. Hinton, P. Dayan, B. Frey, and R. Neal. The wake-sleep algorithm for unsupervised neural networks. *Science*, 268:1158–1161, 1995.

[5] P. R. Montague and T. Sejnowski. The predictive brain: Temporal coincidence and temporal order in synaptic learning mechanisms. *Learning and Memory*, 1:1–33, 1994.

[6] P. Read Montague, Peter Dayan, Christophe Person, and T. Sejnowski. Bee foraging in uncertain environments using predictive hebbian learning. *Nature*, 377:725–728, 1995.

[7] D. Mumford. Neuronal architectures for pattern-theoretic problems. In C. Koch and J. Davis, editors, *Large-scale theories of the cortex*, pages 125–152. MIT Press, 1994.

[8] W. Softky. Could time-series prediction assist visual processing? *Soc. Neurosci. Abstracts*, 21:1499, 1995.

[9] W. Softky. Simple codes vs. efficient codes. *Current Opinion in Neurobiology*, 5:239–247, 1995.

[10] S. Ullman. Sequence-seeking and counterstreams: a model for bidirectional information flow in cortex. In C. Koch and J. Davis, editors, *Large-scale theories of the cortex*, pages 257–270. MIT Press, 1994.

[11] S. Zucker, A. Dobbins, and L. Iverson. Two stages of curve detection suggest two styles of visual computation. *Neural Computation*, 1:68–81, 1989.
